# A Novel Kernel for Learning a Neuron Model from Spike Train Data

**Nicholas Fisher, Arunava Banerjee**
Department of Computer and Information Science and Engineering
University of Florida
Gainesville, FL 32611
`{nfisher,arunava}@cise.ufl.edu`

## Abstract

From a functional viewpoint, a spiking neuron is a device that transforms input spike trains on its various synapses into an output spike train on its axon. We demonstrate in this paper that the function mapping underlying the device can be tractably learned based on input and output spike train data alone. We begin by posing the problem in a classification based framework. We then derive a novel kernel for an $SRM_0$ model that is based on PSP and AHP like functions. With the kernel we demonstrate how the learning problem can be posed as a Quadratic Program. Experimental results demonstrate the strength of our approach.

## 1 Introduction

Neurons are the predominant component of the nervous system and understanding them is a major challenge in modern neuroscience research [1]. Many neuron models have been proposed to understand the dynamics of individual and populations of neurons. Although these models vary in complexity, at a fundamental level they are mechanisms which transform input spike trains into an output spike train. This view has found expression in the Quantitative Single-Neuron Modeling competition where submitted models compete on how accurately they can predict the output spike train of a biological neuron given an input current [2]. Since the vast majority of neurons receive input from chemical synapses [3], a stricter stipulation would be to predict output spikes based on input spike trains at the various synapses of the neuron. There are advantages to this variation of the problem: complicated subthreshold fluctuations in the membrane potential need not be modeled, since models are now judged strictly on the basis of their performance at predicting the timing of output spikes. Models now have the liberty to focus on threshold crossings at the expense of being inaccurate in the subthreshold regime. Not only does the model better represent the *functional* complexity of the input/output transformation of a neuron, comparisons to the real neuron can be conducted in a non-invasive manner.

In this paper we learn a Spike Response Model 0 ($SRM_0$)[4] approximation of a neuron by only considering the timing of all afferent (incoming) and efferent (outgoing) spikes of the neuron over a bounded past. We begin by formulating the problem in a classification based supervised learning framework where spike train data is labeled according to whether the neuron is about to spike, or has recently spiked. We demonstrate that optimizing the model to properly classify this labeled data naturally leads to a quadratic programming problem when combined with an appropriate representation of the model via a dictionary of functions. We then derive a novel *kernel* on spike trains which is computed from a dictionary of post-synaptic potential (PSP) and after-hyperpolarizing potential (AHP) like functions. Finally, experimental results are presented to demonstrate the efficacy of the approach. For a complementary approach to learning a neuron model from spike train data, see [5].

An SRM$_0$ model was chosen for several reasons. First, SRM$_0$ has been shown to be fairly versatile and accurate at modeling biological neurons [6]. Second, SRM$_0$ is a relatively simple neuron model, and therefore is likely to display better generalizability on unseen input. Finally, the disparity between the learned neuron model and the actual neuron could shed light on the various operational modes of biological neurons. It is conceivable that the learned SRM$_0$ model accurately predicts the behavior of the neuron a majority of the time. However, there could be states, bursting for example, where the prediction diverges. In such a case, the neuron can be seen as operating in two different modes, one SRM$_0$ like, and the other not. Multiple models could then be learned to model the neuron in its various operational modes.

## 2 General model of the neuron

It has been shown, that if one assumes a neuron to be a finite precision device with fading memory and a refractory period, then the membrane potential of the neuron, $P$, can be modeled as a function of the timing of the neuron's afferent and efferent spikes which have occurred within a bounded past [7]. Spikes that have aged past this bound, denoted by $\Upsilon$, are considered to have a negligible effect on the present value of $P$. We denote the arrival times of spikes at synapse $j$ using the vector $\mathbf{t}^j = \langle t_1^j, t_2^j \ldots t_{N_j}^j \rangle$, where $N_j$ is bounded from above by the number of spikes that can be present in an $\Upsilon$ window of time. $\mathbf{t}^0$ represents the output spike train of the neuron and vectors $\mathbf{t}^1 \ldots \mathbf{t}^m$ represent spike trains on the input synapses. $t_i^j$ represents the time that has elapsed since that spike was generated or received by the neuron. Spikes are only considered if they occurred within $\Upsilon$ time. We can then formalize the membrane potential function $P : \mathbb{R}^N \to \mathbb{R}$, where $N = \sum_{j=0}^{m} N_j$. $P(\mathbf{t}^0, \ldots, \mathbf{t}^m)$ is defined over the space of all spike trains and reports the present membrane potential of the neuron. The neuron generates a spike when $P(\mathbf{t}^0, \ldots, \mathbf{t}^m) = \Theta$ and $dP/dt \geq 0$, where $\Theta$ is the threshold of the neuron. For notational simplicity, we define the spike configuration, $\mathbf{s} \in \mathbb{R}^N$, which represents the timing of all afferent and efferent spikes within the window of length $\Upsilon$. $\mathbf{s}$ is the vector of vectors, $\mathbf{s} = \langle \mathbf{t}^0, \ldots, \mathbf{t}^m \rangle$. The neuron generates a spike when $P(\mathbf{s}) = \Theta$, $dP/dt \geq 0$.

As discussed in Section 1, we shall learn an SRM$_0$ approximation of the neuron. The SRM$_0$ model uses a bounded past history as described above to calculate the present membrane potential of the neuron. The present membrane potential $\hat{P}$ is calculated as shown in Equation 1. $\eta$ models the effect of a past generated spike, the AHP. $\epsilon_j$ represents the response of the neuron to a presynaptic spike at synapse $j$, the PSP. $u_{\text{rest}}$ is the resting membrane potential. At any given time, the neuron generates a spike if the membrane potential crosses the threshold from below (i.e., $\hat{P}(\mathbf{s}) = \Theta$, $d\hat{P}/dt \geq 0$).

$$\hat{P}(\mathbf{s}) = \sum_{i=1}^{N_0} \eta(t_i^0) + \sum_{j=1}^{m} \sum_{i=1}^{N_j} \epsilon_j(t_i^j) + u_{\text{rest}} \tag{1}$$

## 3 Classification Problem

In order to learn an SRM$_0$ approximation of a neuron in a non-invasive manner, we pose a supervised learning classification problem which labels the given spike train data according to whether the neuron is about to spike or has recently spiked. We denote the former $\mathcal{S}^-$ and the latter $\mathcal{S}^+$. This problem is equivalent to classifying subthreshold spike configurations ($\hat{P}(\mathbf{s}) < \Theta$) from suprathreshold spike configurations ($\hat{P}(\mathbf{s}) \geq \Theta$), which leads to the classification problem shown in Equation 2. It should be noted that the true membrane potential function, $P$, is a feasible solution to this problem since $P(\mathbf{s}) < \Theta \ \forall \mathbf{s} \in \mathcal{S}^-$ and $P(\mathbf{s}) \geq \Theta \ \forall \mathbf{s} \in \mathcal{S}^+$.

$$\text{Min. } \left\| \hat{P}(\mathbf{s}) \right\|^2 \quad \text{s.t. } \hat{P}(\mathbf{s}) - \Theta \geq 1 \ \forall \mathbf{s} \in \mathcal{S}^+ \text{ AND } \hat{P}(\mathbf{s}) - \Theta \leq -1 \ \forall \mathbf{s} \in \mathcal{S}^- \tag{2}$$

To generate training data which belong to $\mathcal{S}^+$ and $\mathcal{S}^-$, we provide the spike configurations which occur at a fixed infinitesimal time differential before and after the neuron generates a spike, as illustrated in Figure 1(a). The spike train at the instant the neuron generated a spike is shown by the solid lines. We shift the spike window infinitesimally into the past (future) to produce a spike configuration $\mathbf{s} \in \mathcal{S}^-(\mathcal{S}^+)$, shown by the up (down) arrows. Notice that the spike which is currently

generated in the output spike train, $\mathbf{t}^0$, emphasized by the dashed circle, is not included in either spike configuration $\mathbf{s}$. The reason it is not included in $\mathbf{s} \in \mathcal{S}^-$ is that it simply has not been generated at that point in time. The reason it is not included in $\mathbf{s} \in \mathcal{S}^+$ is twofold. First, the spike would induce an AHP effect which would cause the membrane potential to fall below the threshold. Second, if it were included, this would cause the classifier to only consider whether or not that particular spike existed when classifying a given spike configuration as a member of $\mathcal{S}^+$ or $\mathcal{S}^-$. If it did exist, it would belong to $\mathcal{S}^+$, and if it did not exist it would belong to $\mathcal{S}^-$. Although this method would work well for the training data, it would not generalize to unseen live spike train data.

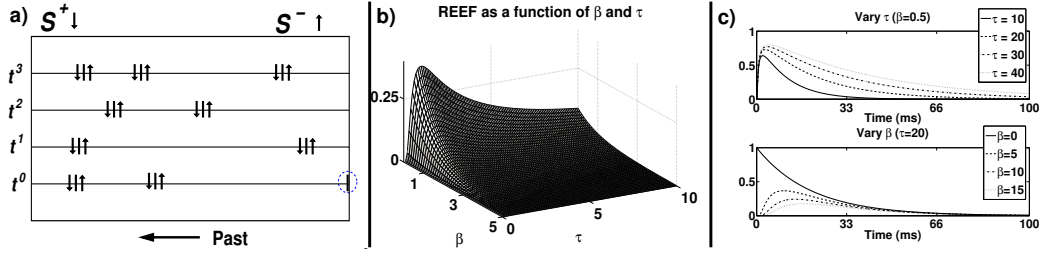

Figure 1: Figure (a) depicts the spike configurations used in the classification problem. Figure (b) shows the REEF for a fixed value of $t = 1$s and variable $\beta$ and $\tau$ values. Figure (c) portrays the form of cross sections of the REEF as a function of $t$ for different values of $\beta$ and $\tau$.

Producing a hypersurface which can separate the supra-threshold spike configurations from the sub-threshold spike configurations *within the spike time feature space*, would be extremely difficult. As discussed above, if we could map a given spike configuration $\mathbf{s}$ to its corresponding membrane potential $P(\mathbf{s})$, then the classification problem is trivial. Although we do not have access to the membrane potential function, we can use a linear combination of functions from a dictionary to reproduce an approximation to the membrane potential function $P$. The choice of the dictionary is crucial. By choosing a dictionary which is tailored to the form of typical PSP and AHP functions, we increase the likelihood of successfully modeling the given neuron.

The $\mathrm{SRM}_0$ model is an additively separable model [8], that is, the membrane potential is a sum of functions of the individual spikes of the spike configuration ($\hat{P}(\mathbf{s}) = \sum_{j=0}^{m} \sum_{i=1}^{N_j} \hat{P}_{ij}(t_i^j)$). This feature lends itself well to modeling the membrane potential using a linear combination of dictionary elements. The dictionary used here was one derived from a function used by MacGregor and Lewis for neuron modeling [9]. It consists of functions (parametrized by $\beta$ and $\tau$) of the form

$$f_{\beta,\tau}(t) = \frac{1}{\tau} \cdot \exp(-\beta/t) \cdot \exp(-t/\tau) \tag{3}$$

We call this the reciprocal exponential – exponential function (REEF) dictionary. Figures 1(b) and (c) present the dictionary for various cross sections of $t$, $\beta$ and $\tau$.

## 4 Approximation of the membrane potential function

We would like to combine members of the chosen dictionary of functions to construct an approximation of the membrane potential function, $P$, which will yield a solution to the classification problem posed in Equation 2. We shall first discuss how this can be achieved in a discrete setting, where we combine a finite number of $\beta$ and $\tau$ parametrized dictionary functions to model $P$. Following this we will discuss a continuous formulation, in which we combine elements drawn from an infinite continuous range of $\beta$ and $\tau$ parametrized dictionary functions to model $P$. In the context of the continuous formulation, we will prove a specific instance of the Representer theorem which was first shown by Kimeldorf and Wahba [10]. The Representer theorem shows that the optimal solution to the posed classification problem must lie in the span of the data points which were used to train the classifier. In the discrete and continuous formulation, we will first model the effect of a single spike for simplicity. We will conclude this section by extending the continuous formulation to the case of multiple spikes on a single synapse, and the case of multiple spikes on multiple synapses.

## 4.1 Discrete Formulation

In the discrete formulation, we wish to approximate the membrane potential function using a linear combination of a finite, predefined set of functions from the REEF dictionary. Focusing on the single spike case, our goal is to model the effect of a single spike on the membrane potential. We denote this effect on the membrane potential by $\hat{P}$ and it is defined as a linear combination of parametrized REEF functions as shown in Equation 4. $f_t(\beta, \tau) = \frac{1}{\tau} \cdot \exp(-\beta/t) \cdot \exp(-t/\tau)$ is now a univariate function over $t$ for fixed values of $\beta$ and $\tau$. A specific set of parameter settings $\{(\beta_1, \tau_1), \ldots, (\beta_M, \tau_1), (\beta_1, \tau_2), \ldots, (\beta_M, \tau_N)\}$ are used to construct a $\hat{P}$ that can best reproduce the effect of the spike on the membrane potential. Inserting Equation 4 into Equation 2 yields a quadratic optimization problem on the mixing coefficients $\alpha_{i,j}$'s.

$$\hat{P}(t) = \sum_{i=1}^{M} \sum_{j=1}^{N} \alpha_{i,j} f_t(\beta_i, \tau_j) \tag{4}$$

The major disadvantage of the discrete formulation is that for any given neuron, the optimal value set of the $\beta$'s and $\tau$'s is unlikely to be known beforehand. While one can argue that the approximation $\hat{P}$ can be improved by increasing $M$ and $N$, as the number of functions increases, so does the dimensionality of the feature space. Since $M$ and $N$ can be increased independent of the size of the training dataset, the procedure is susceptible to over-fitting. To resolve this issue, we shift to a continuous formulation of the problem, which by virtue of the Representer theorem does not suffer from the rising feature space dimensionality issue. The dimensionality of the feature space is now controlled by the span of the training dataset.

## 4.2 Continuous formulation

In the continuous formulation, we consider $\mathcal{L}^2$, the Hilbert space of square integrable functions on the domain $\{\beta, \tau\} \in [0, \infty)^2$. We are concerned with finding a threshold dependent classification function $\hat{P}$, such that $\hat{P}(t) \geq \Theta + 1$ when the spike $t \in \mathcal{S}^+$ and $\hat{P}(t) \leq \Theta - 1$ when $t \in \mathcal{S}^-$. This function is defined in Equation 5.

$$\hat{P}(t) = \langle \alpha(\beta, \tau), f_t(\beta, \tau) \rangle = \int_0^{\infty} \int_0^{\infty} \alpha(\beta, \tau) f_t(\beta, \tau) \, d\beta \, d\tau \tag{5}$$

In this formulation, the *mixing function*, $\alpha(\beta, \tau)$, is by definition a member of $\mathcal{L}^2$. Therefore, if $f_t(\beta, \tau) \in \mathcal{L}^2$, then $\hat{P}(t)$ is finite by the Cauchy-Schwartz inequality since $\langle \alpha(\beta, \tau), f_t(\beta, \tau) \rangle \leq \|\alpha(\beta, \tau)\| \cdot \|f_t(\beta, \tau)\| < \infty$ if both $\|\alpha(\beta, \tau)\| < \infty$ and $\|f_t(\beta, \tau)\| < \infty$. To show that $f_t(\beta, \tau) \in \mathcal{L}^2$ we must show $\langle f_t(\beta, \tau), f_t(\beta, \tau) \rangle < \infty$. For ease of readability we shall henceforth suppress the domain variables in $f_t(\beta, \tau)$ and $\alpha(\beta, \tau)$ and refer to them as $f_t$ and $\alpha$.

### 4.2.1 Proof

$$\langle f_x, f_y \rangle = \int_0^{\infty} \int_0^{\infty} \frac{1}{\tau} \exp\left(-\frac{\beta}{x}\right) \exp\left(-\frac{x}{\tau}\right) \frac{1}{\tau} \exp\left(-\frac{\beta}{y}\right) \exp\left(-\frac{y}{\tau}\right) d\beta d\tau \tag{6}$$

$$= \frac{xy}{(x+y)^2} \tag{7}$$

Therefore $\langle f_t, f_t \rangle = \frac{t \cdot t}{(t+t)^2} = \frac{1}{4} < \infty \; \forall t \in [\epsilon, \infty)$ for some $\epsilon > 0$.

We must note here that by defining the membrane potential function in this manner, we have formulated a problem which yields a solution which is different from the solution to the discrete problem. Since the delta function centered at any arbitrary point $(\beta^*, \tau^*)$ does not belong to $\mathcal{L}^2$, the mixing function $\alpha$ cannot be made up of a linear combination of these delta functions, as is the case in the discrete formulation. In addition, we are not working with a reproducing kernel Hilbert space since we are considering $\mathcal{L}^2$. However, our definition in Equation 5 defines the "point evaluation" of our membrane potential function.

Since $\hat{P}(t)$ is defined using the standard inner product in $\mathcal{L}^2$ with respect to particular members of $\mathcal{L}^2$, we can reformulate the classification problem in Equation 2 as shown in Equation 8. Here $M$ is

the number of data points, $m = 1 \ldots M$, and $y_m$ is the corresponding classification for spike time $t_m$ (that is, $y_m = +1$ if $t_m \in \mathcal{S}^+$ and $y_m = -1$ if $t_m \in \mathcal{S}^-$).

$$\text{Min. } \|\alpha\|^2 \quad \text{s.t. } y_m \left( \langle \alpha, f_{t_m} \rangle - \Theta \right) \geq 1 \quad m = \{1 \ldots M\} \tag{8}$$

We can now use a specific instance of the Representer theorem [10] to show that the optimal solution for $\alpha$ to the optimization problem specified in Equation 8 can be expressed as $\alpha = \sum_{k=1}^{M} \nu_k f_{t_k}$. We can then substitute this equality back into Equation 8 to produce a dual formulation of the optimization problem, which is a standard quadratic programming problem.

### 4.2.2 Representer Theorem

For some $\nu_1, \nu_2, \ldots \nu_M \in \mathbb{R}$, the solution to Equation 8 can be written in the form

$$\alpha = \sum_{k=1}^{M} \nu_k f_{t_k} \tag{9}$$

**Proof**  We consider the subspace of $\mathcal{L}^2$ spanned by the REEF functions evaluated at the times of the given training data points ($\text{span}\{ f_{t_k} : 1 \leq k \leq M \}$). We then consider the projection $\alpha_\parallel$ of $\alpha$ on this subspace. By noting $\alpha = \alpha_\parallel + \alpha_\perp$ and rewriting Equation 8 in its Lagrangian form, we are left with Equation 10. However, by the definition of $\alpha_\perp$, $\langle \alpha_\perp, f_{t_k} \rangle = 0$, which then simplifies the summation term of Equation 10 to only depend upon $\alpha_\parallel$ as shown in Equation 11.

$$\text{Min. } \|\alpha\|^2 + \sum_{k=1}^{M} \lambda_k \left[ 1 - y_k \left( \langle \alpha_\parallel, f_{t_k} \rangle + \langle \alpha_\perp, f_{t_k} \rangle - \Theta \right) \right] \tag{10}$$

$$\text{Min. } \|\alpha\|^2 + \sum_{k=1}^{M} \lambda_k \left[ 1 - y_k \left( \langle \alpha_\parallel, f_{t_k} \rangle - \Theta \right) \right] \tag{11}$$

In addition, by considering the relation shown in Equation 12, we find that the first term is minimized when $\alpha = \alpha_\parallel$. Hence, the optimal solution to Equation 8 will lie in the aforementioned subspace and therefore have the form of Equation 9.

$$\|\alpha\|^2 = \|\alpha_\parallel\|^2 + \|\alpha_\perp\|^2 \geq \|\alpha_\parallel\|^2 \tag{12}$$

### 4.2.3 Dual Representation

We can now substitute the form of the optimal solution shown in Equation 9 back into the original optimization problem shown in Equation 8. This leads to the problem in Equation 13 which is equivalent to Equation 14. The resultant quadratic programming problem is solvable given that we have access to the positive definite matrix $K$, which was derived in Section 4.2.1 and is shown in Equation 15.

$$\text{Min. } \left\| \sum_{k=1}^{M} \nu_k f_{t_k} \right\|^2 \quad \text{s.t. } y_m \left( \left\langle \sum_{k=1}^{M} \nu_k f_{t_k}, f_{t_m} \right\rangle - \Theta \right) \geq 1 \quad m = \{1 \ldots M\} \tag{13}$$

$$\text{Min. } \sum_{i=1}^{M} \sum_{j=1}^{M} \nu_i \nu_j K(t_i, t_j) \quad \text{s.t. } y_m \left( \sum_{k=1}^{M} \nu_k K(t_k, t_m) - \Theta \right) \geq 1 \quad m = \{1 \ldots M\} \tag{14}$$

$$K(t_i, t_j) = \langle f_{t_i}, f_{t_j} \rangle = \int_0^\infty \int_0^\infty f_{t_i} f_{t_j} \, d\beta \, d\tau = \frac{t_i t_j}{(t_i + t_j)^2} \tag{15}$$

### 4.3 Single Synapse

We are now in a position to extend the framework to multiple spikes on a single synapse. Since we are learning an $\text{SRM}_0$ approximation of a neuron, we assume that the effects of spikes are additively separable [8] and that each spike's effect on the membrane potential for the given synapse is *identical*. Introducing the latter assumption is the core contribution of this section. We first define the threshold dependent classification function for a single spike in a manner identical to that of the single spike formulation shown in Equation 5. This will be the "stereotyped" effect that a spike arriving at this synapse has on the membrane potential. Note that the AHP effect of the output spike train can be modeled seamlessly (as a virtual synapse) in this framework.

### 4.3.1 Primal Problem

We now consider the additive effects of multiple spikes arriving at a synapse. We define the vector $\mathbf{t}^m = \langle t_1^m, t_2^m, \ldots, t_{N_m}^m \rangle$ to be the $m^{th}$ data point, which consists of $N_m$ spikes, represented by their spike times. Note that we have abused notation. Instead of the superscript repeatedly referring to the synapse in question, it now refers to the data point. The primal optimization problem, defined in Equation 16, is equivalent to Equation 17.

$$\text{Min. } \|\alpha\|^2 \text{ s.t. } y_m \left( \sum_{h=1}^{N_m} \langle \alpha, f_{t_h^m} \rangle - \Theta \right) \geq 1 \quad m = \{1 \ldots M\} \tag{16}$$

$$\text{Min. } \|\alpha\|^2 \text{ s.t. } y_m \left( \left\langle \alpha, \sum_{h=1}^{N_m} f_{t_h^m} \right\rangle - \Theta \right) \geq 1 \quad m = \{1 \ldots M\} \tag{17}$$

The Representer theorem states that the optimal $\alpha$ must lie in $\text{span}\{\sum_{i=1}^{N_k} f_{t_i^k} : 1 \leq k \leq M \}$. We omit the formal proof since it follows along the lines of the previous case. Therefore, the optimal $\alpha$ to Equation 17 will be of the form

$$\alpha = \sum_{k=1}^{M} \nu_k \sum_{i=1}^{N_k} f_{t_i^k} \tag{18}$$

### 4.3.2 Dual Problem

Substituting back Equation 18 yields the dual problem Equation 19, which can be solved given the positive definite kernel in Equation 20.

$$\text{Min. } \left\| \sum_{k=1}^{M} \nu_k \sum_{i=1}^{N_k} f_{t_i^k} \right\|^2 \text{ s.t. } y_m \left( \left\langle \sum_{k=1}^{M} \nu_k \sum_{i=1}^{N_k} f_{t_i^k}, \sum_{h=1}^{N_m} f_{t_h^m} \right\rangle - \Theta \right) \geq 1 \quad m = \{1 \ldots M\} \tag{19}$$

$$K(\mathbf{t}^p, \mathbf{t}^q) = \left\langle \sum_{i=1}^{N_p} f_{t_i^p}, \sum_{k=1}^{N_q} f_{t_k^q} \right\rangle = \sum_{i=1}^{N_p} \sum_{k=1}^{N_q} \left\langle f_{t_i^p}, f_{t_k^q} \right\rangle = \sum_{i=1}^{N_p} \sum_{k=1}^{N_q} \frac{t_i^p \cdot t_k^q}{(t_i^p + t_k^q)^2} \tag{20}$$

## 4.4 Multiple Synapses

In the multiple synapse case, the principles are identical to that of the single synapse, with the exception that spikes arriving at different synapses could have different effects on the membrane potential, depending on the strength/type of the synaptic junction. Therefore, we keep the effects of each synapse on the membrane potential separate by assigning each synapse its own $\alpha$ function.

### 4.4.1 Primal Problem

Since each synapse and the output has its own $\alpha$ function, this simply adds another summation term over the $S$ synapses and the output (indexed by 0). The primal optimization problem is defined in Equation 21 which is equivalent to Equation 22. $S$ is the number of synapses, $N_{m,s}$ is the number of spikes on the $s^{th}$ synapse of the $m^{th}$ data point, and $t_h^{m,s}$ is the timing of the $h^{th}$ spike on the $s^{th}$ synapse of the $m^{th}$ data point.

$$\text{Min. } \sum_{s=0}^{S} \|\alpha_s\|^2 \text{ s.t. } y_m \left( \sum_{s=0}^{S} \sum_{h=1}^{N_{m,s}} \left\langle \alpha_s, f_{t_h^{m,s}} \right\rangle - \Theta \right) \geq 1 \quad m = \{1 \ldots M\} \tag{21}$$

$$\text{Min. } \sum_{s=0}^{S} \|\alpha_s\|^2 \text{ s.t. } y_m \left( \sum_{s=0}^{S} \left\langle \alpha_s, \sum_{h=1}^{N_{m,s}} f_{t_h^{m,s}} \right\rangle - \Theta \right) \geq 1 \quad m = \{1 \ldots M\} \tag{22}$$

The Representer theorem states that the optimal $\alpha_s$ for the $s^{th}$ synapses must lie in $\text{span}\{\sum_{i=1}^{N_{k,s}} f_{t_i^{k,s}} : 1 \leq k \leq M\}$. This is identical to the single synapse case for each synapse,

and therefore, the optimal $\alpha_s$ to Equation 22 will be of the form

$$\alpha_s = \sum_{k=1}^{M} \nu_k \sum_{i=1}^{N_{k,s}} f_{t_i^{k,s}} \tag{23}$$

### 4.4.2 Dual Problem

Substituting Equation 23 into Equation 22 yields the dual problem shown in Equation 24 which can be solved given access to the positive definite kernel defined in Equation 25.

$$\text{Min.} \qquad \sum_{s=0}^{S} \left\| \sum_{k=1}^{M} \nu_k \sum_{i=1}^{N_{k,s}} f_{t_i^{k,s}} \right\|^2 \tag{24}$$

$$\text{s.t.} \qquad y_m \left( \sum_{s=0}^{S} \left\langle \sum_{k=1}^{M} \nu_k \sum_{i=1}^{N_{k,s}} f_{t_i^{k,s}}, \sum_{h=1}^{N_{m,s}} f_{t_h^{m,s}} \right\rangle - \Theta \right) \geq 1 \quad m = \{1 \ldots M\}$$

$$K(\mathbf{t}^p, \mathbf{t}^q) = \sum_{s=0}^{S} \left\langle \sum_{i=1}^{N_{p,s}} f_{t_i^{p,s}}, \sum_{k=1}^{N_{q,s}} f_{t_k^{q,s}} \right\rangle = \sum_{s=0}^{S} \sum_{i=1}^{N_{p,s}} \sum_{k=1}^{N_{q,s}} \frac{t_i^{p,s} \cdot t_k^{q,s}}{(t_i^{p,s} + t_k^{q,s})^2} \tag{25}$$

### 4.5 Summary

With the above kernels we are able to formulate quadratic programming problems which can be solved with SVM$^{\text{light}}$ [11]. The choice of the dictionary used to derive the kernel is critical to the success of this technique. A dictionary of functions tailored to the forms of PSPs and AHPs will perform better than a more general class of functions. The properties of the REEF dictionary which make it suitable for this problem are its exponential decay as well as its additive separability [8]. This explains why a Gaussian radial basis function (GRBF) does not work well for this problem. The GRBF kernel is not additive. A slight variation of the GRBF which takes the sum of Gaussian functions, rather than their product, was also explored. This performed better than the GRBF; however it did not perform well when applied to more complicated neurons.

## 5 Results

To test the kernel we learned SRM$_0$ neurons with increasing levels of complexity. We first considered a simplistic neuron which only received spikes on a single synapse. We then increased the complexity of the neuron, by introducing AHP effects as well as different types (excitatory and inhibitory) of afferent synapses with varying synaptic weights. The PSP effect was modeled via the classical alpha function $[\text{PSP}(t) = C \cdot t \cdot \exp(-t/\tau)]$ while the AHP effect was modeled by an exponential function$[\text{AHP}(t) = K \cdot \exp(-t/\tau)]$. Although we learned neurons with varying complexity, for want of space, we discuss here the case of a single neuron that received input spike trains from $4$ excitatory synapses and $1$ inhibitory synapse to mimic the ratio of connections observed in the cortex [12]. The stereotyped PSP for the excitatory and inhibitory synapses differed in their rise and fall times. The parameters for the stereotyped PSP were set as follows. For the excitatory PSP, $C = 0.1$ and $\tau = 10$, where $t$ is in units of milliseconds. For the inhibitory PSP, $C = -0.39$ and $\tau = 5$. For the AHP, $K = -16.667$ and $\tau = 2$.

We first trained the classifier using 100,000 seconds of spike train data. Only the spike configurations occurring at fixed differentials before and after the neuron emitted a spike were considered. The input spike trains were generated using an inhomogeneous Poisson process, where the rate was varied sinusoidally around the intended mean spike rate in order to produce a more general set of training data. This resulted in 1,647,249 training data points, however only 10,681 of them were used in the solution as support vectors. After training, we tested our model using 100 seconds of unseen data. All spike configurations were considered when testing, regardless of temporal proximity to spike generation. To quantify our results, we first calculated the accuracy $\left( \frac{\text{correct classifications}}{\text{total data points}} \right)$, the sensitivity $\left( \frac{\text{correct positive classifications}}{\text{total positive data points}} \right)$, and specificity $\left( \frac{\text{correct negative classifications}}{\text{total negative data points}} \right)$.

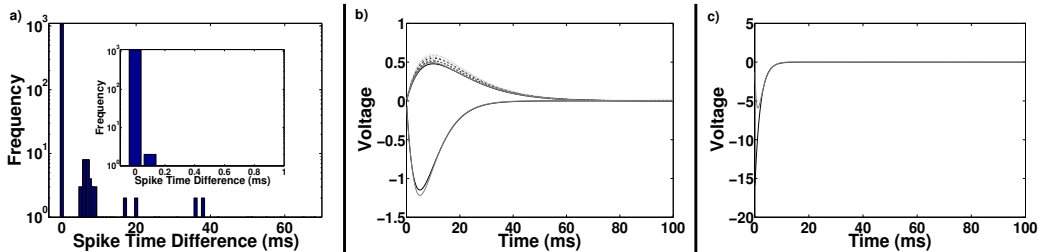

Figure 2: Figure (a) shows histograms of the difference in time between the actual and predicted spike time by the learned model. Figure (b) shows the various PSP approximations (gray) in comparison to the PSP functions used by the neuron (black). Figure (c) depicts the AHP approximation (gray) and the AHP function used by the neuron (black).

They were $0.9947$, $0.9532$ and $0.9948$ respectively. We also calculated a histogram of how close the spike predictions were. For every spike produced by the neuron, we determined the temporal proximity of the closest spike time predicted by the model. We then histogrammed this data. Figure 2(a) shows two histograms depicting these calculations. The larger histogram contains predictions with time differences varying between 0 and 70 ms, with a bin size of 1 ms while the inlaid histogram ranges from 0 to 10 ms and has a bin size of 0.1 ms. Both use a logarithmic scale on the y-axis. From the histograms, we see that the vast majority of spikes were predicted correctly (with a temporal proximity of 0 ms) and that out of the mispredicted spike times, the temporal proximity of all predicted spikes fell within 70 ms of the actual spike time.

In Figures 2(b) and 2(c) we display a comparison of the approximated PSP and AHP versus the true PSP and AHP. To calculate the classification model's approximated PSP we artificially send a single spike across each input synapse. We artificially generate a spike to produce the AHP approximation. By considering the distance of the single spike data point from the classifier's margin as the spike ages, we can get a scaled and translated version of the PSP and AHP. The figures show these approximations scaled and translated back appropriately. In Figure 2(b) we show the approximations of the PSPs for the input synapses. The approximations are shown in gray; the true PSPs are shown in black. The different line styles are representative of the different synapses and therefore have varying synaptic weights. A similar image for the AHP is shown in Figure 2(c). We note that there are small differences between the approximated and the true functions. If the PSP and AHP approximations were exact, we would have seen perfect classification results. However, as with most machine learning techniques, the quality of the solution is limited by the training data given.

## 6   Conclusion

In this paper we have developed a classification framework which uses a novel kernel derived from a REEF dictionary to produce an $SRM_0$ approximation of a neuron. The technique used is non-invasive in the sense that it only requires the timing of afferent and efferent spikes within a certain bounded past. The REEF dictionary was chosen due to its similarity to PSP and AHP functions used in a neuron model proposed by MacGregor and Lewis [9].

By producing an $SRM_0$ approximation, which is additively separable [8], we produce a model which is both versatile and accurate [6]. In addition, it is a relatively simple model, which allows for increased generalizability to unseen input. The simplicity of the $SRM_0$ model has the potential to allow us to observe deviations between the model and the neuron, which can lead to insights on the various behavioral modes of neurons.

**Acknowledgments**

This work was supported by a National Science Foundation grant (NSF IIS-0902230) to A.B.

## References

[1] R. Jolivet, A. Roth, F. Schürmann, W. Gerstner, and W. Senn. Special issue on quantitative neuron modeling. *Biological Cybernetics*, 99(4):237–239, 2008.

[2] W. Gerstner and R. Naud. How Good Are Neuron Models? *Science*, 326(5951):379–380, 2009.

[3] W. Gerstner and W. Kistler. *Spiking Neuron Models: An Introduction*. Cambridge University Press New York, NY, USA, 2002.

[4] R. Jolivet, T.J. Lewis, and W. Gerstner. The spike response model: a framework to predict neuronal spike trains. *Artificial Neural Networks and Neural Information Processing–ICANN/ICONIP 2003*, pages 173–173, 2003.

[5] L. Paninski, J.W. Pillow, and E.P. Simoncelli. Maximum likelihood estimation of a stochastic integrate-and-fire neural encoding model. *Neural Computation*, 16(12):2533–2561, 2004.

[6] R. Jolivet, T.J. Lewis, and W. Gerstner. Generalized integrate-and-fire models of neuronal activity approximate spike trains of a detailed model to a high degree of accuracy. *Journal of Neurophysiology*, 92(2):959–976, 2004.

[7] A. Banerjee. On the phase-space dynamics of systems of spiking neurons. I: Model and experiments. *Neural Computation*, 13(1):161–193, 2001.

[8] Tadeusz Stanisz. Functions with separated variables. Master's thesis, Zeszyty Naukowe Uniwerstyetu Jagiellonskiego, 1969.

[9] R.J. MacGregor and E.R. Lewis. *Neural Modeling*. Plenum Press, New York, 1977.

[10] G. Kimeldorf and G. Wahba. Some results on Tchebycheffian spline functions. *Journal of Mathematical Analysis and Applications*, 33(1):82–95, 1971.

[11] T. Joachims. Making large-scale support vector machine learning practical. In *Advances in Kernel Methods*, pages 169–184. MIT Press, 1999.

[12] E.M. Izhikevich. Simple model of spiking neurons. *IEEE Transactions on Neural Networks*, 14(6):1569–1572, 2003.

